# An Autonomous Robotic System
# For Mapping Abandoned Mines

**D. Ferguson**[1]**, A. Morris**[1]**, D. Hähnel**[2]**, C. Baker**[1]**, Z. Omohundro**[1]**, C. Reverte**[1]
**S. Thayer**[1]**, C. Whittaker**[1]**, W. Whittaker**[1]**, W. Burgard**[2]**, S. Thrun**[3]

| [1]The Robotics Institute | [2]Computer Science Department | [3]Computer Science Department |
| Carnegie Mellon University | University of Freiburg | Stanford University |
| Pittsburgh, PA | Freiburg, Germany | Stanford, CA |

## Abstract

We present the software architecture of a robotic system for mapping abandoned mines. The software is capable of acquiring consistent 2D maps of large mines with many cycles, represented as Markov random fields. 3D C-space maps are acquired from local 3D range scans, which are used to identify navigable paths using A* search. Our system has been deployed in three abandoned mines, two of which inaccessible to people, where it has acquired maps of unprecedented detail and accuracy.

## 1 Introduction

This paper describes the navigation software of a deployed robotic system for mapping subterranean spaces such as abandoned mines. Subsidence of abandoned mines poses a major problem for society, as do ground water contaminations, mine fires, and so on. Most abandoned mines are inaccessible to people, but some are accessible to robots. Autonomy is a key requirement for robots operating in such environments, due to a lack of wireless communication technology for subterranean spaces.

Our vehicle, shown in Figure 1 (see [1] for a detailed hardware description) is equipped with two actuated laser range finders. When exploring and mapping unknown mines, it alternates short phases of motion guided by 2D range scans, with phases in which the vehicle rests to acquire 3D range scans. An analysis of the 3D scans leads to a path that is then executed, using rapidly acquired 2D scans to determine the robot's motion relative to the 3D map. If no such path is found a high-level control module adjusts the motion direction accordingly.

Acquiring consistent large-scale maps without external geo-referencing through GPS is largely considered an open research issue. Our approach relies on efficient statistical techniques for generating such maps in real-time. At the lowest level, we employ a fast scan matching algorithm for registering successive scans, thereby recovering robot odometry. Groups of scans are then converted into local maps, using Markov random field representations (MRFs) to characterize the residual path uncertainty. Loop closure is attained by adding constraints into those MRFs, based on a maximum likelihood (ML) estimator. However, the brittleness of the ML approach is overcome by a "lazy" data association mechanism that can undo and redo past associations so as to maximize the overall map consistency.

To navigate, local 3D scans are mapped into $2\frac{1}{2}$D terrain maps, by analyzing surface gradients and vertical clearance in the 3D scans. The result is subsequently transformed into cost

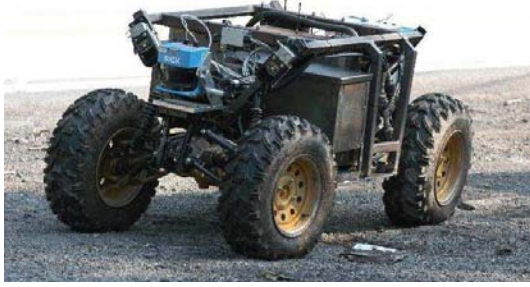

**Figure 1**: The Groundhog robot is a 1,500 pound custom-built vehicle equipped with onboard computing, laser range sensing, gas and sinkage sensors, and video recording equipment. Its purpose is to explore and map abandoned mines.

functions expressed in the robot's three-dimensional configuration space, by convolving the $2\frac{1}{2}$D terrain maps with kernels that describe the robot's footprints in different orientations. Fast A* planning is then employed in configuration space to generate paths executed through PD control.

The system has been tested in a number of mines. Some of the results reported here were obtained via manual control in mines accessible to people. Others involved fully autonomous exploration, for which our robot operated fully self-guided for several hours beyond the reach of radio communication.

## 2  2D Mapping

### 2.1  Generating Locally Consistent Maps

As in [6, 9], we apply an incremental scan matching technique for registering scans, acquired using a forward-pointed laser range finder while the vehicle is in motion. This algorithm aligns scans by iteratively identifying nearby points in pairs of consecutive range scans, and then calculating the relative displacement and orientation of these scans by minimizing the quadratic distance of these pairs of points [2]. This approach leads to the recovery of two quantities: locally consistent maps and an estimate of the robot's motion. It is well-understood [3, 6], however, that local scan matching is incapable of achieving globally consistent maps. This is because of the residual error in scan matching, which accumulates over time. The limitation is apparent in the map shown in Figure 2a, which is the result of applying local scan matching in a mine that is approximately 250 meters wide.

Our approach addresses this problem by explicitly representing the uncertainty in the map and the path using a Markov random field (MRF) [11]. More specifically, the data acquired through every five meters of consecutive robot motion is mapped into a local map [3]. Figure 3a shows such a local map. The absolute location of orientation of the $k$-th map will be denoted by $\xi_k = (\begin{array}{ccc} x_k & y_k & \theta_k \end{array})^T$; here $x$ and $y$ are the Cartesian coordinates and $\theta$ is the orientation. From the scan matcher, we can retrieve relative displacement information of the form $\delta_{k,k-1} = (\begin{array}{ccc} \Delta x_{k,k-1} & \Delta y_{k,k-1} & \Delta\theta_{k,k-1} \end{array})^T$ which, if scan matching was error-free, would enable us to recover absolute information via the following recursion (under the boundary condition $\xi_0 = (0,0,0)^T$)

$$\xi_k \;\; = \;\; f(\xi_{k-1}, \delta_{k,k-1}) \;\; = \;\; \left( \begin{array}{c} x_{k-1} + \Delta x_{k,k-1}\cos\theta_{k,k-1} + \Delta y_{k,k-1}\sin\theta_{k-1} \\ y_{k-1} - \Delta x_{k,k-1}\sin\theta_{k,k-1} + \Delta y_{k,k-1}\cos\theta_{k-1} \\ \theta_{k-1} + \Delta\theta_{k,k-1} \end{array} \right) \!\!(1)$$

However, scan matching is not without errors. To account for those errors, our approach generalizes this recursion into a Markov random field (MRF), in which each variable $\Xi = \xi_1, \xi_2, \ldots$ is a (three-dimensional) node. This MRF is defined through the potentials:

$$\phi(\xi_k, \xi_{k-1}) \;\; = \;\; \exp -\tfrac{1}{2}(\xi_k - f(\xi_{k-1}, \delta_{k,k-1}))^T R_{k,k-1}(\xi_k - f(\xi_{k-1}, \delta_{k,k-1})) \;\; (2)$$

Here $R_{k,k-1}$ is the inverse covariance of the uncertainty associated with the transition $\delta_{k,k-1}$. Since the MRF is a linear chain without cycles, the mode of this MRF is the solution to the recursion defined in (1). Figure 3b shows the MRF for the data collected in the

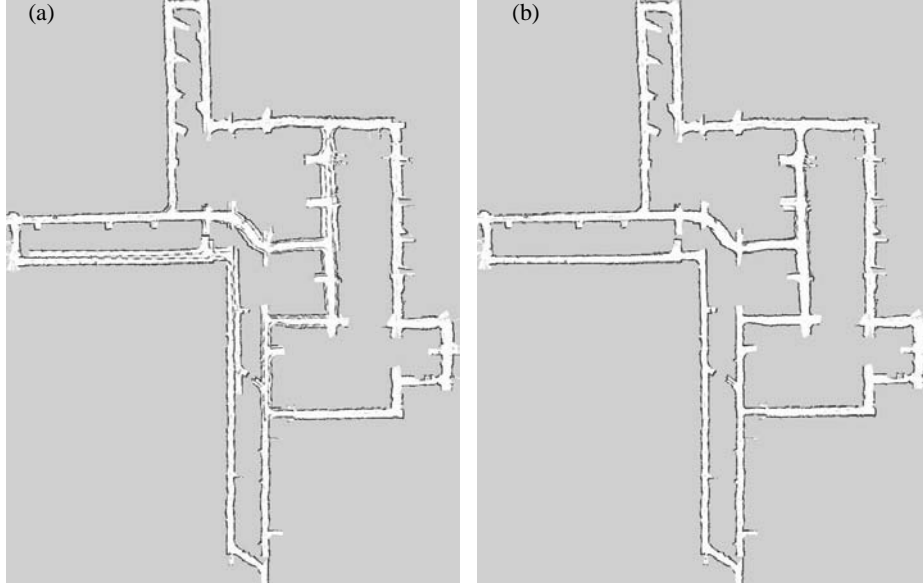

(a)                                                    (b)

**Figure 2**: Mine map with incremental ML scan matching (left) and using our lazy data association approach (right). The map is approximately 250 meters wide.

Bruceton Research Mine, over a distance of more than a mile. We note this representation generalizes the one in [11], who represent posteriors by a local bank of Kalman filters.

## 2.2   Enforcing Global Consistency

The key advantage of the MRF representation is that it encompasses the residual uncertainty in local scan matching. This enables us to alter the shape of the map in accordance with global consistency constraints. These constraints are obtained by matching local maps acquired at different points in time (e.g., when closing a large cycle). In particular, if the $k$-th map overlaps with some map $j$ acquired at an earlier point in time, our approach localizes the robot relative to this map using once again local scan matching. As a result, it recovers a relative constraint $\phi(\xi_k, \xi_j)$ between the coordinates of non-adjacent maps $\xi_k$ and $\xi_j$. This constraint is of the same form as the local constraints in (2), hence is represented by a potential. For any fixed set of such potentials $\Phi = \{\phi(\xi_k, \xi_j)\}$, the resulting MRF is described through the following negative log-likelihood function

$$-\log p(\Xi) \quad = \quad \text{const.} + \tfrac{1}{2} \sum_{k,j} (\xi_k - f(\xi_j, \delta_{k,j}))^T \, R_{k,j} \, (\xi_k - f(\xi_j, \delta_{k,j})) \qquad (3)$$

where $\Xi = \xi_1, \xi_2, \dots$ is the set of all map poses, and $f$ is defined in (1).

Unfortunately, the resulting MRF is *not* a linear chain any longer. Instead, it contains cycles. The variables $\Xi = \xi_1, \xi_2, \dots$ can be recovered using any of the standard inference algorithms for inference on graphs with cycles, such as the popular loopy belief propagation algorithm and related techniques [5, 14, 17]. Our approach solves this problem by matrix inversion. In particular, we linearize the function $f$ using a Taylor expansion:

$$f(\xi_j, \delta_{k,j}) \quad \approx \quad f(\bar{\xi}_j) + F_{k,j}(\xi_j - \bar{\xi}_j) \qquad (4)$$

where $\bar{\xi}_j$ denotes a momentary estimate of the variables $\xi_j$ (e.g., the solution of the recursion (1) without the additional data association constraints). The matrix $F_{k,j} =$

(a)                          (b)

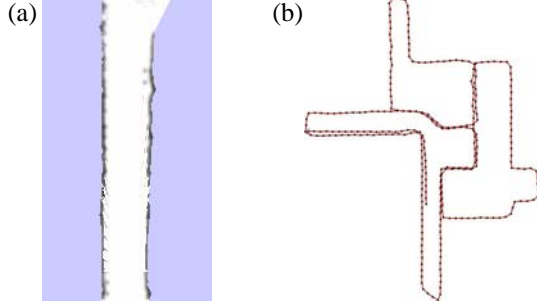

**Figure 3**: (a) Example of a local map. (b) The Markov random field: Each node is the center of a local map, acquired when traversing the Bruceton Research Mine near Pittsburgh, PA.

$\nabla_{\xi_j} f(\bar{\xi}_j, \delta_{k,j})$ is the Jacobean of $f(\xi_j, \delta_{k,j})$ at $\bar{\xi}_j$:

$$F_{k,j}x \;=\; \begin{pmatrix} 1 & 0 & -\Delta x_{k,j}\sin\bar{\theta}_k + \Delta y_{k,j}\cos\bar{\theta}_k \\ 0 & 1 & -\Delta x_{k,j}\cos\bar{\theta}_k - \Delta y_{k,j}\sin\bar{\theta}_k \\ 0 & 0 & 1 \end{pmatrix} \tag{5}$$

The resulting negative log-likelihood is given by

$$-\log p(\Xi) \;\approx\; \text{const.} + \tfrac{1}{2}\sum_{k,j}(\xi_k - f(\bar{\xi}_j) - F_{k,j}(\xi_j - \bar{\xi}_j))^T \; \sigma_{k,j}^{-1} \; (\xi_k - f(\bar{\xi}_j) - F_{k,j}(\xi_j - \bar{\xi}_j))$$

is quadratic in the variables $\Xi$ of the form $\text{const.} + (A\Xi - a)^T\, R\, (A\Xi - a)$, where $A$ is a diagonal matrix, $a$ is a vector, and $R$ is a sparse matrix that is non-zero for all elements $j, k$ in the set of potentials. The minimum of this function is attained at $(A^T R A)^{-1} A^T R a$. This solution requires the inversion of a sparse matrix. Empirically, we find that this inversion can be performed very efficiently using an inversion algorithm described in [15]; it only requires a few seconds for matrices composed of hundreds of local map positions (and it appears to be numerically more stable than the solution in [11, 6]). Iterative application of this linearized optimization quickly converges to the mode of the MRF, which is the set of locations and orientations $\Xi$. However, we conjecture that recent advances on inference in loopy graphs can further increase the efficiency of our approach.

### 2.3 Lazy Data Association Search

Unfortunately, the approach described thus far leads only to a consistent map when the additional constraints $\phi(\xi_k, \xi_j)$ obtained after loop closure are correct. These constraints amount to a maximum likelihood solution for the challenging data association problem that arises when closing a loop. When loops are large, this ML solution might be wrong—a problem that has been the source of an entire literature on SLAM (simultaneous localization and mapping) algorithms. Figure 4a depicts such a situation, obtained when operating our vehicle in a large abandoned mine.

The current best algorithms apply proactive particle filter (PF) techniques to solve this problem [4, 8, 12, 13]. PF techniques sample from the path posterior. When closing a loop, random variations in these samples lead to different loop closures. As long as the correct such closure is in the set of surviving particle filters, the correct map can be recovered. In the context of our present system, this approach suffers from two disadvantages: it is computationally expensive due to its proactive nature, and it provides no mechanism for recovery should the correct loop closure not be represented in the particle set.

Our approach overcomes both of these limitations. When closing a loop, it always picks the most likely data association. However, it also provides a mechanism to undo and redo past data association decisions. The exact data association algorithm involves a step that monitors the likelihood of the most recent sensor measurement given the map. If this likelihood falls below a threshold, data association constraints are recursively undone and replaced by other constraints of decreasing likelihood (including the possibility of not generating a constraint at all). The search terminates if the likelihood of the most recent measurement

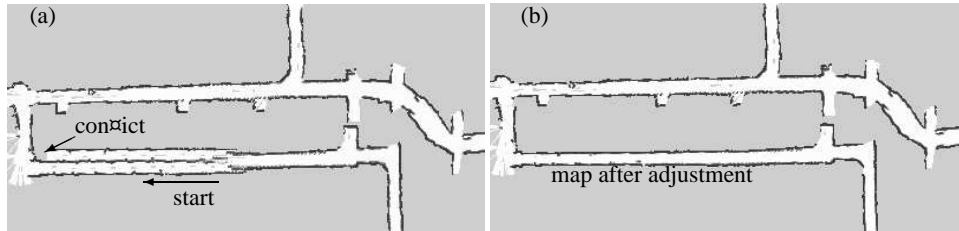

(a)                                                                (b)

conflict

start                                    map after adjustment

**Figure 4**: Example of our lazy data association technique: When closing a large loop, the robot first erroneously assumes the existence of a second, parallel hallway. However, this model leads to a gross inconsistency as the robot encounters a corridor at a right angle. At this point, our approach recursively searches for improved data association decisions, arriving at the map shown on the right.

exceeds the threshold [7]. In practice, the threshold test works well, since global inconsistencies tend to induce gross inconsistencies in the robot's measurements at some point in time.

The algorithm is illustrated in Figure 4. The left panel shows the ML association after traversing a large loop inside a mine: At first, it appears that the existence of two adjacent corridors is more likely than a single one, according to the estimated robot motion. However, as the robot approaches a turn, a noticeable inconsistency is detected. Inconsistencies are found by monitoring the measurement likelihood, using a threshold for triggering an exception. As a result, our data association mechanism recursively removes past data association constraints back to the most recent loop closure, and then "tries" the second most likely hypothesis. The result of this backtracking step is shown in the right panel of Figure 4. The backtracking requires a fraction of a second, and with high likelihood leads to a globally consistent map and, as a side-effect, to an improved estimate of the map coordinates $\Xi$. Figure 2b shows a proto-typical corrected map, which is globally consistent.

## 3   Autonomous Navigation

2D maps are sufficient for localizing robots inside mines; however, they are insufficient to navigate a robot due to the rugged nature of abandoned mines. Our approach to navigation is based on 3D maps, acquired in periodic intervals while the vehicle suspends motion to scan its environment. A typical 3D scan is shown in Figure 5a; others are shown in Figure 7.

### 3.1   $2\frac{1}{2}$D Terrain Maps

In a first processing step, the robot projects local 3D maps onto $2\frac{1}{2}$D terrain maps, such as the one shown in Figure 5b. The gray-level in this map illustrates the degree at which the map is traversable: the brighter a 2D location, the better suited it is for navigation.

The terrain map is obtained by analyzing all measurements $\langle x, y, z \rangle$ in the 3D scan (where $z$ is the vertical dimension). For each rectangular surface region $\{x_{\min}; x_{\max}\} \times \{y_{\min}; y_{\max}\}$, it identifies the minimum $z$-value, denoted $\underline{z}$. It then searches for the largest $z$ value in this region whose distance to $\underline{z}$ does not exceed the vehicle height (plus a safety margin); this value will be called $\bar{z}$. The difference $\bar{z} - \underline{z}$ is the navigational coefficient: it loosely corresponds to the ruggedness of the terrain under the height of the robot. If no measurement is available for the target region $\{x_{\min}; x_{\max}\} \times \{y_{\min}; y_{\max}\}$, the region is marked as unknown. For safety reasons, multiple regions $\{x_{\min}; x_{\max}\} \times \{y_{\min}; y_{\max}\}$ overlap when building the terrain map. The terrain map is subsequently convolved with a narrow radial kernel that serves as a repellent potential field, to keep the robot clear of obstacles.

### 3.2   Configuration Space Maps

The terrain map is used to construct a collection of maps that describe the robot's configuration space, or C-space [10]. The C-space is the three-dimensional space of poses that

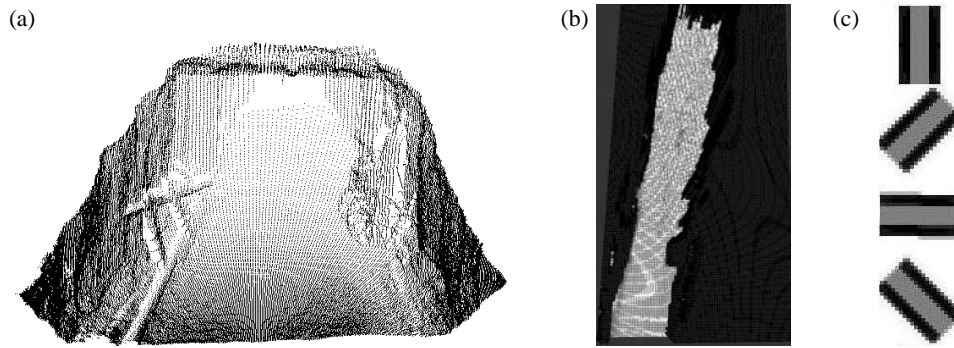

(a)                  (b)          (c)

**Figure 5**: (a) A local 3D model of the mine corridor, obtained by a scanning laser range finder. (b) The corresponding $2\frac{1}{2}$D terrain map extracted from this 3D snapshot: the brighter a location, the easier it is to navigate. (c) Kernels for generating directional C-space maps from the $2\frac{1}{2}$D terrain map. The two black bars in each kernel correspond to the vehicle's tires. Planning in these C-space maps ensures that the terrain under the tires is maximally navigable.

the vehicle can assume; it comprises the $x$-$y$ location along with the vehicle's orientation $\theta$. The C-space maps are obtained by convolving the terrain map with oriented kernels that describe the robot's footprint. Figure 5c shows some of these kernels: Most value is placed in the wheel area of the vehicle, with only a small portion assigned to the area in between, where the vehicle's clearance is approximately 30 centimeters. The intuition of using such a kernel is as follows: Abandoned mines often possess railroad tracks, and while it is perfectly acceptable to navigate with a track between the wheels, traversing or riding these tracks causes unnecessary damage to the tires and will increase the energy consumption. The result of this transformation is a collection of C-space maps, each of which applies to a different vehicle orientation.

### 3.3 Corridor Following

Finally, A* search is employed in C-space to determine a path to an unexplored area. The A* search is initiated with an array of goal points, which places the highest value at locations at maximum distance straight down a mine corridor. This approach finds the best path to traverse, and then executes it using a PD controller.

If no such path can be found even within a short range (2.5 meters), the robot decides that the hallway is not navigable and initiates a high-level decision to turn around. This technique has been sufficient for our autonomous exploration runs thus far (which involved straight hallway exploration), but it does not yet provide a viable solution for exploring multiple hallways connected by intersections (see [16] for recent work on this topic).

## 4 Results

The approach was tested in multiple experiments, some of which were remotely operated while in others the robot operated autonomously, outside the reach of radio communication. On October 27, 2002, Groundhog was driven under manual control into the Florence Mine near Burgettstown, PA. Figure 6b shows a picture of the tethered and remotely controlled vehicle inside this mine, which has not been entered by people for many decades. Its partially flooded nature prevented an entry into the mine for more than approximately 40 meters. Maps acquired in this mine are shown in Figure 9.

On May 30, 2003, Groundhog successfully explored an abandoned mine using the fully autonomous mode. The mine, known as the Mathies Mine near Pittsburgh, is part of a large mine system near Courtney, PA. Existing maps for this mine are highly inaccurate, and the conditions inside the mine were unknown to us. Figure 6a shows the robot as it enters the mine, and Figure 7a depicts a typical 3D scan acquired in the entrance area.

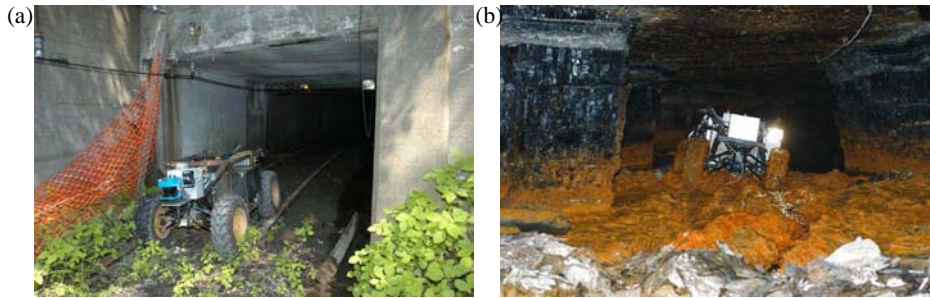

**Figure 6**: (a) The vehicle as it enters the Mathies Mine on May 30, 2003. It autonomously descended 308 meters into the mine before making the correct decision to turn around due to a blockage inside the mine. (b) The vehicle, as it negotiates acidic mud under manual remote control approximately 30 meters into the Florence Mine near Burgettstown, PA.

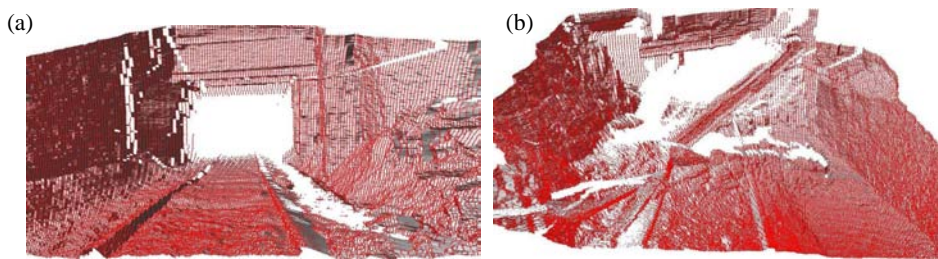

**Figure 7**: 3D local maps: (a) a typical corridor map that is highly navigable. (b) a map of a broken ceiling bar that renders the corridor segment unnavigable. This obstacle was encountered 308 meters into the abandoned Mathies Mine.

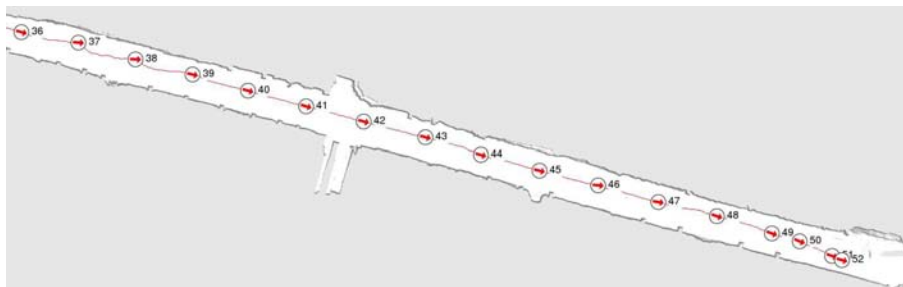

**Figure 8**: Fraction of the 2D mine map of the Mathies Mine, autonomously explored by the Ground-hog vehicle. Also shown is the path of the robot and the locations at which it chose to take 3D scans. The protruding obstacle shows up as a small dot-like obstacle in the 2D map.

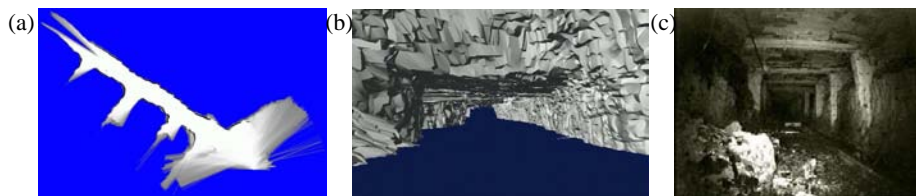

**Figure 9**: (a) A small 2D map acquired by Groundhog in the Florence Mine near Burgettstown, PA. This remotely-controlled mission was aborted when the robot's computer was flooded by water and mud in the mine. (b) View of a local 3D map of the ceiling. (c) Image acquired by Groundhog inside the Mathies Mine (a dry mine).

After successfully descending 308 meters into the Mathies Mine, negotiating some rough terrain along the way, the robot encountered a broken ceiling beam that draped diagonally across the robot's path. The corresponding 3D scan is shown in Figure 7b: it shows rubble on the ground, along with the ceiling bar and two ceiling cables dragged down by the bar. The robot's A* motion planner failed to identify a navigable path, and the robot made the appropriate decision to retreat. Figure 8 shows the corresponding 2D map; the entire map is 308 meters long, but here we only show the final section, along with the path and the location at which the robot stop to take a 3D scan. An image acquired in this mine is depicted in Figure 9c.

## 5 Conclusion

We have described the software architecture of a deployed system for robotic mine mapping. The most important algorithmic innovations of our approach are new, lazy techniques for data association, and a fast technique for navigating rugged terrain. The system has been tested under extreme conditions, and generated accurate maps of abandoned mines inaccessible to people.

## Acknowledgements

We acknowledge the contributions of the students of the class 16865 Mobile Robot Development at CMU who helped build Groundhog. We also acknowledge the assistance provided by Bruceton Research Mine (Paul Stefko), MSHA, PA-DEP, Workhorse Technologies, and the various people in the mining industry who supported this work. Finally, we also gratefully acknowledge financial support by DARPA's MARS program.

## References

[1] C. Baker, Z. Omohundro, S. Thayer, W. Whittaker, M. Montemerlo, and S. Thrun. A case study in robotic mapping of abandoned mines. *FSR-03*.

[2] P. Besl and N. McKay. A method for registration of 3d shapes. *PAMI* 14(2), 1992.

[3] M. Bosse, P. Newman, M. Soika, W. Feiten, J. Leonard, and S. Teller. An atlas framework for scalable mapping. *ICRA-03*.

[4] A. Eliazar and R. Parr. DP-SLAM: Fast, robust simultaneous localization and mapping without predetermined landmarks. *IJCAI-03*.

[5] Anshul Gupta, George Karypis, and Vipin Kumar. Highly scalable parallel algorithms for sparse matrix factorization. *Trans. Parallel and Distrib. Systems*, 8(5), 1997.

[6] J.-S. Gutmann and K. Konolige. Incremental mapping of large cyclic environments. CIRA-00.

[7] D. Hähnel, W. Burgard, B. Wegbreit, and S. Thrun. Towards lazy data association in SLAM. *11th International Symposium of Robotics Research*, Sienna, 2003.

[8] D. Hähnel, D. Fox, W. Burgard, and S. Thrun. A highly efficient FastSLAM algorithm for generating cyclic maps of large-scale environments from raw laser range measurements. Submitted to *IROS-03*.

[9] D. Hähnel, D. Schulz, and W. Burgard. Map building with mobile robots in populated environments. *IROS-02*.

[10] J.-C. Latombe. *Robot Motion Planning*. Kluwer, 1991.

[11] F. Lu and E. Milios. Globally consistent range scan alignment for environment mapping. *Autonomous Robots*: 4, 1997.

[12] M. Montemerlo, S. Thrun, D. Koller, and B. Wegbreit. FastSLAM 2.0: An improved particle filtering algorithm for simultaneous localization and mapping that provably converges. *IJCAI-03*.

[13] K. Murphy. Bayesian map learning in dynamic environments. *NIPS-99*.

[14] K.P. Murphy, Y. Weiss, and M.I. Jordan. Loopy belief propagation for approximate inference: An empirical study. *UAI-99*

[15] W. H. Press. *Numerical recipes in C: the art of scientific computing*. Cambridge Univ. Press, 1988.

[16] R. Simmons, D. Apfelbaum, W. Burgard, M. Fox, D. an Moors, S. Thrun, and H. Younes. Coordination for multi-robot exploration and mapping. *AAAI-00*.

[17] M. J. Wainwright. *Stochastic processes on graphs with cycles: geometric and variational approaches*. PhD thesis, MIT, 2002.
